# 488 Solutions to the XOR Problem

**Frans M. Coetzee** *
*coetzee@ece.cmu.edu*
Department of Electrical Engineering
Carnegie Mellon University
Pittsburgh, PA 15213

**Virginia L. Stonick**
*ginny@ece.cmu.edu*
Department of Electrical Engineering
Carnegie Mellon University
Pittsburgh, PA 15213

## Abstract

A globally convergent homotopy method is defined that is capable of sequentially producing large numbers of stationary points of the multi-layer perceptron mean-squared error surface. Using this algorithm large subsets of the stationary points of two test problems are found. It is shown empirically that the MLP neural network appears to have an extreme ratio of saddle points compared to local minima, and that even small neural network problems have extremely large numbers of solutions.

## 1  Introduction

The number and type of stationary points of the error surface provide insight into the difficulties of finding the optimal parameters of the network, since the stationary points determine the degree of the system[1]. Unfortunately, even for the small canonical test problems commonly used in neural network studies, it is still unknown how many stationary points there are, where they are, and how these are divided into minima, maxima and saddle points.

Since solving the neural equations explicitly is currently intractable, it is of interest to be able to numerically characterize the error surfaces of standard test problems. To perform such a characterization is non-trivial, requiring methods that reliably converge and are capable of finding large subsets of distinct solutions. It can be shown[2] that methods which produce only one solution set on a given trial become inefficient (at a factorial rate) at finding large sets of multiple distinct solutions, since the same solutions are found repeatedly. This paper presents the first provably globally convergent homotopy methods capable of finding large subsets of the

Currently with Siemens Corporate Research, Princeton NJ 08540

stationary points of the neural network error surface. These methods are used to empirically quantify not only the number but also the type of solutions for some simple neural networks.

## 1.1 Sequential Neural Homotopy Approach Summary

We briefly acquaint the reader with the principles of homotopy methods, since these approaches differ significantly from standard descent procedures.

Homotopy methods solve systems of nonlinear equations by mapping the known solutions from an initial system to the desired solution of the unsolved system of equations. The basic method is as follows: Given a *final* set of equations $f(x) = 0, x \in D \subseteq \Re^n$ whose solution is sought, a *homotopy* function $h : D \times T \to \Re^n$ is defined in terms of a parameter $\tau \in T \subset \Re$, such that

$$h(x, \tau) = \begin{cases} g(x) & \text{when } \tau = 0 \\ f(x) & \text{when } \tau = 1 \end{cases}$$

where the *initial system* of equations $g(x) = 0$ has a known solution. For optimization problems $f(x) = \nabla_x \epsilon^2(x)$ where $\epsilon^2(x)$ is the error measure. Conceptually, $h(x, \tau) = 0$ is solved numerically for $x$ for increasing values of $\tau$, starting at $\tau = 0$ at the known solution, and incrementally varying $\tau$ and correcting the solution $x$ until $\tau = 1$, thereby tracing a path from the initial to the final solutions.

The power and the problems of homotopy methods lie in constructing a suitable function $h$. Unfortunately, for a given $f$ most choices of $h$ will fail, and, with the exception of polynomial systems, no guaranteed procedures for selecting $h$ exist. Paths generally do not connect the initial and final solutions, either due to non-existence of solutions, or due to paths diverging to infinity. However, if a theoretical proof of existence of a suitable trajectory can be constructed, well-established numerical procedures exist that reliably track the trajectory.

The following theorem, proved in [2], establishes that a suitable homotopy exists for the standard feed-forward backpropagation neural networks:

**Theorem 1.1** *Let $\epsilon^2$ be the unregularized mean square error (MSE) problem for the multi-layer perceptron network, with weights $\beta \in \Re^n$. Let $\beta_0 \in U \subset \Re^n$ and $a \in V \subset \Re^n$, where $U$ and $V$ are open bounded sets. Then except for a set of measure zero $(\beta, a) \in U \times V$, the solutions $(\beta, \tau)$ of the set of equations*

$$h(\beta, \tau) = (1 - \tau)(\beta - \beta_0) + \tau D_\beta \left(\epsilon^2 + \mu\psi(||\beta - a||^2)\right) = 0 \qquad (1)$$

*where $\mu > 0$ and $\psi : \Re \to \Re$ satisfies $2\psi''(\alpha^2)\alpha^2 + \psi'(\alpha^2) > 0$ as $\alpha \to \infty$, form non-crossing one dimensional trajectories for all $\tau \in \Re$, which are bounded $\forall \tau \in [0, 1]$. Furthermore, the path through $(\beta_0, 0)$ connects to at least one solution $(\beta^*, 1)$ of the regularized MSE error problem*

$$\min_\beta \left(\epsilon^2 + \mu\psi(||\beta - a||^2)\right) \qquad (2)$$

On $\tau \in [0, 1]$ the approach corresponds to a pseudo-quadratic error surface being deformed continuously into the final neural network error surface[1]. Multiple solu-

tions can be obtained by choosing different initial values $\beta_0$. Every desired solution $\beta^*$ is accessible via an appropriate choice of $a$, since $\beta_0 = \beta^*$ suffices.

Figure 1 qualitatively illustrates typical paths obtained for this homotopy[2]. The paths typically contain only a few solutions, are disconnected and diverge to infinity. A novel two-stage homotopy[2, 3] is used to overcome these problems by constructing and solving *two* homotopy equations. The first homotopy system is as described above. A synthetic second homotopy solves an auxiliary set of equations on a non-Euclidean compact manifold ($S^n(0; R) \times \Lambda$, where $\Lambda$ is a compact subset of $R$) and is used to move between the disconnected trajectories of the first homotopy. The method makes use of the topological properties of the compact manifold to ensure that the secondary homotopy paths do not diverge.

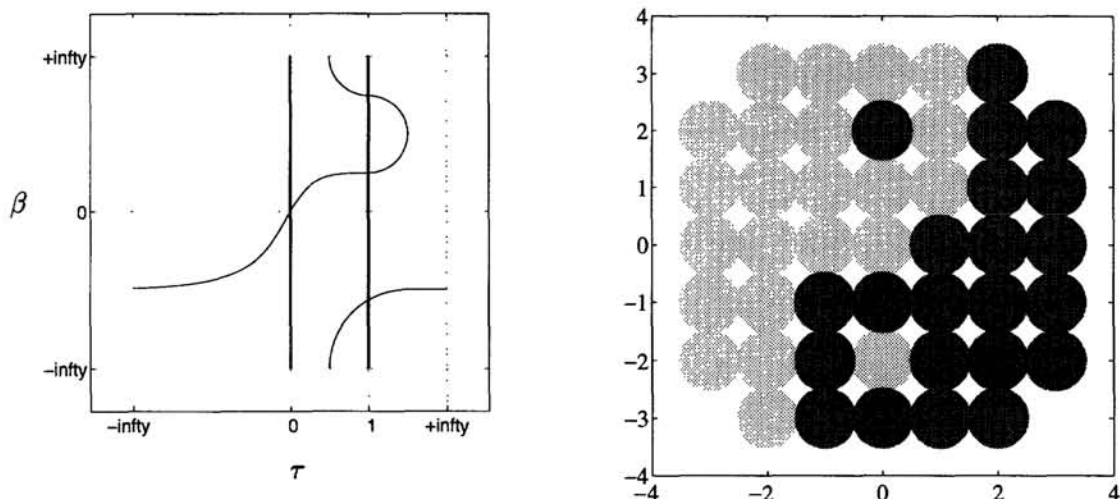

Figure 1: (a) Typical homotopy trajectories, illustrating divergence of paths and multiple solutions occurring on one path. (b) Plot of two-dimensional vectors used as training data for the second test problem (Yin-Yang problem).

## 2  Test Problems

The test problems described in this paper are small to allow for (i) a large number of repeated runs, and (ii) to make it possible to numerically distinguish between solutions. Classification problems were used since these present the only interesting small problems, even though the MSE criterion is not necessarily best for classification. Unlike most classification tasks, all algorithms were forced to approximate the stationary point accurately by requiring the $l_1$ norm of the gradient to be less than $10^{-10}$, and ensuring that solutions differed in $l_1$ by more than 0.01.

The historical XOR problem is considered first. The data points $(-1, -1)$, $(1, 1)$, $(-1, 1)$ and $(1, -1)$ were trained to the target values $-0.8, -0.8, 0.8$ and $0.8$. A network with three inputs (one constant), two hidden layer nodes and one output node were used, with hyperbolic tangent transfer functions on the hidden and final

nodes. The regularization used $\mu = 0.05$, $\psi(x) = x$ and $a = 0$ (no bifurcations were found for this value during simulations). This problem was chosen since it is small enough to serve as a benchmark for comparing the convergence and performance of the different algorithms. The second problem, referred to as the Yin-Yang problem, is shown in Figure 1. The problem has 23 and 22 data points in classes one and two respectively, and target values $\pm 0.7$. Empirical evidence indicates that the smallest single hidden layer network capable of solving the problem has five hidden nodes. We used a net with three inputs, five hidden nodes and one output. This problem is interesting since relatively high classification accuracy is obtained using only a single neuron, but a 100% classification performance requires at least five hidden nodes and one of only a few global weight solutions.

The stationary points form equivalence classes under renumbering of the weights or appropriate interchange of weight signs. For the XOR problem each solution class contains up to $2^2\ 2! = 8$ distinct solutions; for the Yin-Yang network, there are $2^5\ 5! = 3840$ symmetries. *The equivalence classes are reported in the following sections.*

## 3   Test Results

A Ribak-Poliere conjugate gradient (CG) method was used as a control since this method can find only minima, as contrasted to the other algorithms, all of which are attracted by all stationary points. In the second algorithm, the homotopy equation (1) was solved by following the main path until divergence. A damped Newton (DN) method and the two-stage homotopy method completed the set of four algorithms considered. The different algorithms were initialized with the same $n$ random weights $\beta_0 \in S^{n-1}(0; \sqrt{2}n)$.

### 3.1   Control - The XOR problem

The total number and classification of the solutions obtained for 250 iterations on each algorithm are shown in Table 1.

Table 1: Number of equivalence class solutions obtained. XOR Problem

| Algorithm | # Solutions | #Minima | # Maxima | #Saddle Points |
|---|---|---|---|---|
| CG | 17 | 17 | 0 | 0 |
| DN | 44 | 6 | 0 | 38 |
| One Stage | 28 | 16 | 0 | 12 |
| Two Stage | 61 | 17 | 0 | 44 |
| Total Distinct | 61 | 17 | 0 | 44 |

The probability of finding a given solution on a trial is shown in Figure 2. The two-stage homotopy method finds almost every solution from every initial point. In contrast to the homotopy approaches, the Newton method exhibits poor convergence, even when heavily damped. The sets of saddle points found by the DN algorithm and the homotopy algorithms are to a large extent disjoint, even though the same initial weights were used. For the Newton method solutions close to the initial point are typically obtained, while the initial point for the homotopy algo-

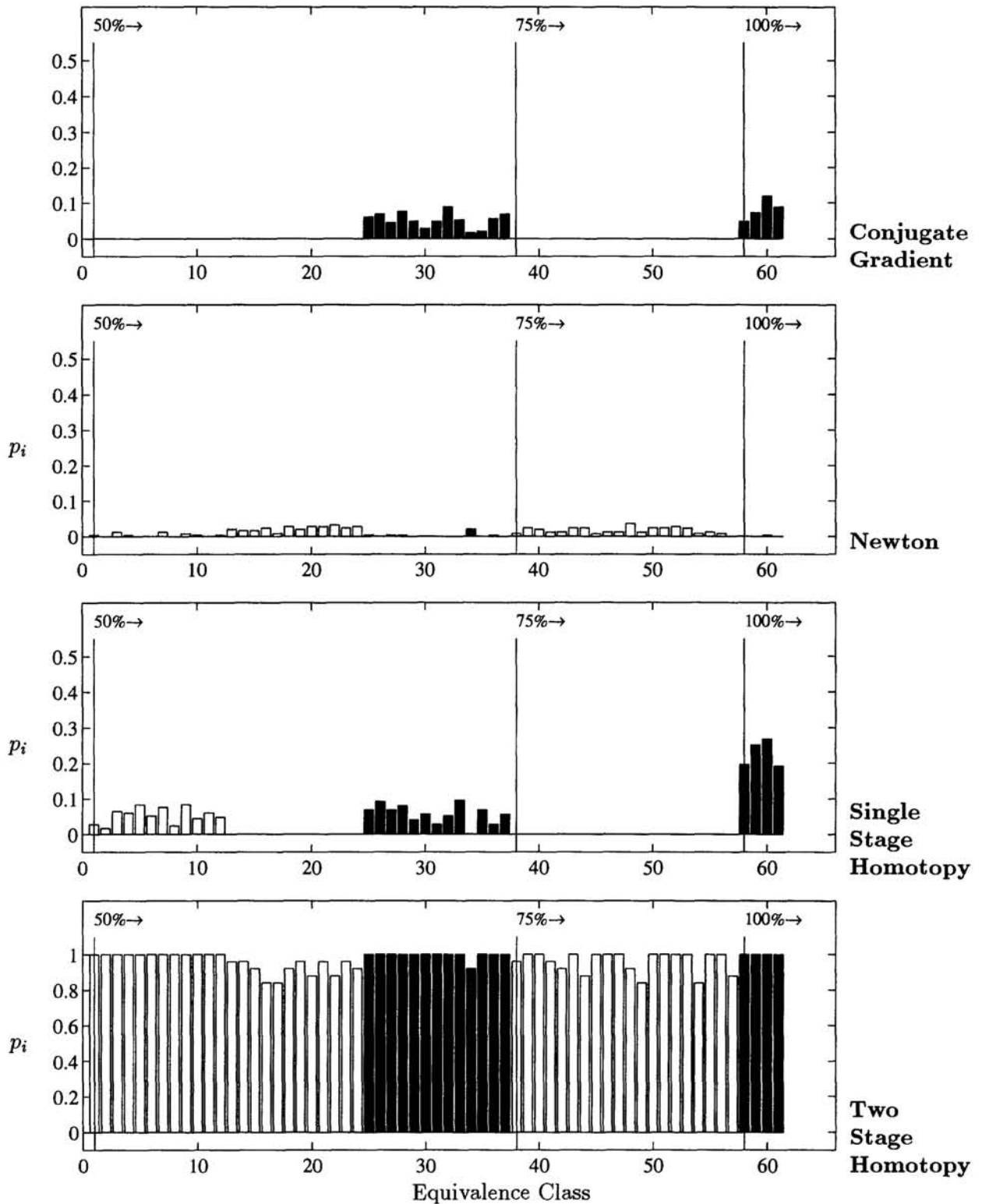

Figure 2: Probability of finding equivalence class $i$ on a trial. Solutions have been sorted based on percentage of the training set correctly classified. Dark bars indicate local minima, light bars saddle points. XOR problem

Table 2: Number of solutions correctly classifying $x$% of target data.

| Classification | 25 % | 50 % | 75 % | 100 % |
|---|---|---|---|---|
| Minimum | 17 | 17 | 4 | 4 |
| Saddle | 44 | 44 | 20 | 0 |
| Total Distinct | 61 | 61 | 24 | 4 |

rithm might differ significantly from the final solution. This difference illustrates that homotopy arrives at solutions in a fundamentally different way than descent approaches.

Based on these results we conclude that the two-stage homotopy meets its objective of significantly increasing the number of solutions produced on a single trial. The homotopy algorithms converge more reliably than Newton methods, in theory and in practise. These properties make homotopy attractive for characterizing error surfaces. Finally, due to the large number of trials and significant overlap between the solution sets for very different algorithms, we believe that Tables 1-2 represent accurate estimates for the number and types of solutions to the regularized XOR problem.

## 3.2   Results on the Yin-Yang problem

The first three algorithms for the Yin-Yang problem were evaluated for 100 trials. The conjugate gradient method showed excellent stability, while the Newton method exhibited serious convergence problems, even with heavy damping. The two-stage algorithm was still producing solutions when the runs were terminated after multiple weeks of computer time, allowing evaluation of only ten different initial points.

Table 3: Number of equivalence class solutions obtained. Yin-Yang Problem

| Algorithm | # Solutions | #Minima | # Maxima | #Saddle Points |
|---|---|---|---|---|
| Conjugate Gradient | 14 | 14 | 0 | 0 |
| Damped Newton | 10 | 0 | 0 | 10 |
| One Stage Homotopy | 78 | 15 | 0 | 63 |
| Two Stage Homotopy | 1633 | 12 | 0 | 1621 |
| Total Distinct | 1722 | 28 | 0 | 1694 |

Table 4: Number of solutions correctly classifying $x$% of target data.

| Classification | 75 | 80 | 90 | 95 | 96 | 97 | 98 | 99 | 100 % |
|---|---|---|---|---|---|---|---|---|---|
| Minimum | 28 | 28 | 28 | 26 | 26 | 5 | 5 | 2 | 2 |
| Saddle | 1694 | 1694 | 1682 | 400 | 400 | 13 | 13 | 3 | 3 |
| Total Distinct | 1722 | 1722 | 1710 | 426 | 426 | 18 | 18 | 5 | 5 |

The results in Tables 3-4 for the number of minima are believed to be accurate, due to verification provided by the conjugate gradient method. The number of saddle

points should be seen as a *lower bound*. The regularization ensured that the saddle points were well conditioned, *i.e.* the Hessian was not rank deficient, and these solutions are therefore distinct point solutions.

## 4   Conclusions

The homotopy methods introduced in this paper overcome the difficulties of poor convergence and the problem of repeatedly finding the same solutions. The use of these methods therefore produces significant new empirical insight into some extraordinary unsuspected properties of the neural network error surface.

The error surface appears to consist of relatively few minima, separated by an extraordinarily large number of saddle points. While one recent paper by Goffe *et al* [4] had given some numerical estimates based on which it was concluded that a large number of minima in neural nets exist (they did not find a significant number of these), this extreme *ratio* of saddle points to minima appears to be unexpected. No maxima were discovered in the above runs; in fact none appear to exist within the sphere where solutions were sought (this seems likely given the regularization). The numerical results reveal astounding complexity in the neural network problem. If the equivalence classes are complete, then 488 solutions for the XOR problem are implied, of which 136 are minima. For the Yin-Yang problem, 6,600,000+ solutions and 107,250+ minima were characterized. For the simple architectures considered, these numbers appear extremely high. We are unaware of any other system of equations having these remarkable properties.

Finally, it should be noted that the large number of saddle points and the small ratio of minima to saddle points in neural problems can create tremendous computational difficulties for approaches which produce stationary points, rather than simple minima. The efficiency of any such algorithm at producing solutions will be negated by the fact that, from an optimization perspective, most of these solutions will be useless.

**Acknowledgements.** The partial support of the National Science Foundation by grant MIP-9157221 is gratefully acknowledged.

## Footnotes

[1] The common engineering heuristic whereby some arbitrary error surface is relaxed into another error surface generally does *not* yield well defined trajectories.

[2]Note that the homotopy equation and its trajectories exist outside the interval $\tau = [0, 1]$.

## References

[1] E. H. Rothe, *Introduction to Various Aspects of Degree Theory in Banach Spaces.* Mathematical Surveys and Monographs (23), Providence, Rhode Island: American Mathematical Society, 1986. ISBN 0-82218-1522-9.

[2] F. M. Coetzee, *Homotopy Approaches for the Analysis and Solution of Neural Network and Other Nonlinear Systems of Equations.* PhD thesis, Carnegie Mellon University, Pittsburgh, PA, May 1995.

[3] F. M. Coetzee and V. L. Stonick, "Sequential homotopy-based computation of multiple solutions to nonlinear equations," in *Proc. IEEE ICASSP*, (Detroit), IEEE, May 1995.

[4] W. L. Goffe, G. D. Ferrier, and J. Rogers, "Global optimization of statistical functions with simulated annealing," *Jour. Econometrics*, vol. 60, no. 1-2, pp. 65–99, 1994.